# Unsupervised context sensitive language acquisition from a large corpus

**Zach Solan, David Horn, Eytan Ruppin**
Sackler Faculty of Exact Sciences
Tel Aviv University
Tel Aviv, Israel 69978
{*rsolan,horn,ruppin*}*@post.tau.ac.il*

**Shimon Edelman**
Department of Psychology
Cornell University
Ithaca, NY 14853, USA
*se37@cornell.edu*

## Abstract

We describe a pattern acquisition algorithm that learns, in an unsupervised fashion, a streamlined representation of linguistic structures from a plain natural-language corpus. This paper addresses the issues of learning structured knowledge from a large-scale natural language data set, and of generalization to unseen text. The implemented algorithm represents sentences as paths on a graph whose vertices are words (or parts of words). Significant patterns, determined by recursive context-sensitive statistical inference, form new vertices. Linguistic constructions are represented by trees composed of significant patterns and their associated equivalence classes. An input module allows the algorithm to be subjected to a standard test of English as a Second Language (ESL) proficiency. The results are encouraging: the model attains a level of performance considered to be "intermediate" for 9th-grade students, despite having been trained on a corpus (CHILDES) containing transcribed speech of parents directed to small children.

## 1 Introduction

A central tenet of generative linguistics is that extensive innate knowledge of grammar is essential to explain the acquisition of language from positive-only data [1, 2]. Here, we explore an alternative hypothesis, according to which syntax is an abstraction that emerges from exposure to language [3], coexisting with the corpus data within the same representational mechanism. Far from parsimonious, the representation we introduce allows partial overlap of linguistic patterns or constructions [4]. The incremental process of acquisition of patterns is driven both by structural similarities and by statistical information inherent in the data, so that frequent strings of similar composition come to be represented by the same pattern. The degree of abstraction of a pattern varies: it may be high, as in the case of a frame with several slots, each occupied by a member of an equivalence class associated with it, or low, as in the extreme case of idioms or formulaic language snippets, where there is no abstraction at all [5, 6]. The acquired patterns represent fully the original data, and, crucially, enable structure-sensitive generalization in the production and the assimilation of unseen examples.

Previous approaches to the acquisition of linguistic knowledge, such as $n$-gram Hidden

Markov Models (HMMs) that use raw data, aimed not at grammar induction but rather at expressing the probability of a sentence in terms of the conditional probabilities of its constituents. In comparison, statistical grammar induction methods aim to identify the most probable grammar, given a corpus [7, 8]. Due to the difficulty of this task, a majority of such methods have focused on supervised learning [9]. Grammar induction methods that do attempt unsupervised learning can be categorized into two classes: those that use corpora tagged with part-of-speech information, and those that work with raw, untagged data. The former includes such recent work as alignment-based learning [10], regular expression ("local grammar") extraction [11], and algorithms that rely on the Minimum Description Length (MDL) principle [12].

The present work extends an earlier study [13] which offered preliminary results demonstrating the feasibility of unsupervised learning of linguistic knowledge from raw data. Here, we describe a new learning model and its implementation and extensive testing on a large corpus of transcribed spoken language from the CHILDES collection [14] (the larger corpora used in many other computational studies do not focus on children-directed language). Our new results suggest that useful patterns embodying syntactic and semantic knowledge of language can indeed be extracted from untagged corpora in an unsupervised manner.

## 2   The ADIOS model

The ADIOS (Automatic DIstillation Of Structure) model has two components: (1) a Representational Data Structure (RDS) graph, and (2) a Pattern Acquisition (PA) algorithm that progressively refines the RDS in an unsupervised fashion. The PA algorithm aims to detect *significant patterns* (SP): similarly structured sequences of primitives that recur in the corpus. Each SP has an associated *equivalence class* (EC), which is a set of alternative primitives that may fit into the slot in the SP to construct a given path through the graph (see Figure 1a). The manner whereby the model supports generalization is exemplified in Figure 1c. The algorithm requires neither prior classification of the primitives into syntactic categories, nor even a pre-setting of their scope: it can bootstrap itself from a corpus in which all the words have been broken down into their constituent characters.

One of the few free parameters in the earlier version of the model, ADIOS1, was the length $L$ of the typical pattern the system was expected to acquire. Although presetting the value of $L$ sufficed to learn simple artificial grammars, it proved to be problematic for natural language corpora. On the one hand, a small value of $L$ led to over-generalization, because of insufficient uniformity of ECs associated with short SPs (not enough context sensitivity). On the other hand, using large values of $L$ in conjunction with the ADIOS1 statistical learning algorithm did not lead to the emergence of well-supported SPs. The ADIOS2 model addresses this issue by first identifying long significant paths (SPATH) in the graph, then analyzing their $k$-gram statistics to identify short significant patterns SP.

### 2.1   Step 1: identifying a significant path

For each $path_i$ (sequence of elements $e_1 \rightarrow e_2 \rightarrow \ldots \rightarrow e_k$) longer than a given threshold, the algorithm constructs a set $\mathcal{P} = \{p_1, \ldots, p_m\}$ of paths of the same length as $path_i$. Each of the paths in $\mathcal{P}(path_i)$ consists of the same non-empty prefix (some sequence of graph edges), an equivalence class of vertices, and the same non-empty suffix (another sequence of edges); as an example, consider the set of three paths starting with 'is' and ending with the end of sentence symbol 'END' in Figure 1. Each such set is assigned a score $S(\mathcal{P}) \doteq \sum_j s(path_j)$, with $s(\cdot)$ defined by eq. 1. This score assesses the likelihood that $\mathcal{P}$ captures a significant regularity rather than a random fluctuation in the data. The set with the maximal score in a given pass over the corpus is the SPATH.

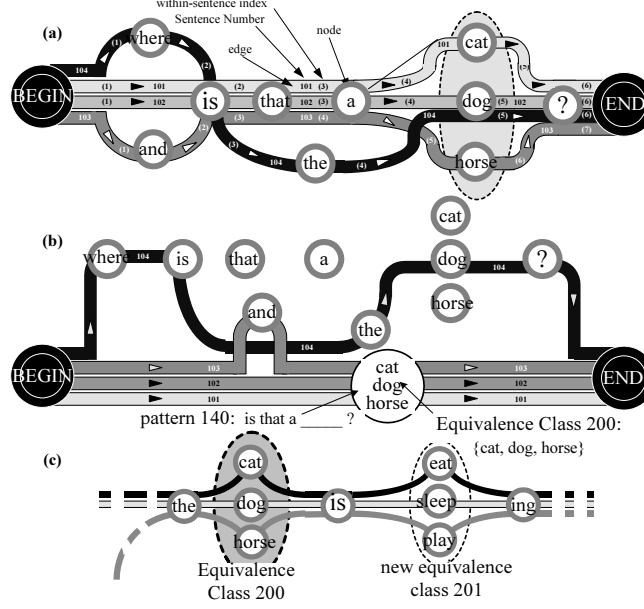

Figure 1: (a) A small portion of the RDS, which is a directed multi-graph, for a simple corpus containing sentences #101 (is that a cat?) #102 (is that a dog?) #103 (and is that a horse?) #104 (where is the dog?). Each sentence is depicted by a solid colored line; edge direction is marked by arrows and is labeled by the sentence number and within-sentence index. The sentences in this example join a pattern is that a {dog, cat, horse} ?. (b). The abstracted pattern and the equivalence class associated with it are highlighted (edges that belong to sequences not subsumed by this pattern, e.g., #104, are untouched). (c) The identification of new significant patterns is done using the acquired equivalence classes (e.g., #200). In this manner, the system "bootstraps" itself, recursively distilling more and more complex patterns. This kind of abstraction also supports generalization: the original three sentences (shaded paths) form a pattern with two equivalence classes, which can then potentially generate six new sentences (e.g., the cat is play-ing and the horse is eat-ing).

$$s(path_i) = P^{(k)}(path_i) \log \left( P^{(k)}(path_i)/P^{(2)}(path_i) \right) \tag{1}$$

$$P^{(k)}(path_i) = P(e_1)P(e_2|e_1)P(e_3|e_1 \to e_2)...P(e_k|e_1 \to e_2 \to ... \to e_{k-1}) \tag{2}$$

$$P^{(2)}(path_i) = P(e_1)P(e_2|e_1)P(e_3|e_2)...P(e_k|e_{k-1}) \tag{3}$$

The algorithm estimates the probabilities of different paths from the respective $k$-gram statistics ($k$ being the length of the paths in the set under consideration), as per eq. 2. We observe that $P^{(1)}(path_i)$ corresponds to the "first order" probability of choosing the set of nodes $e_1, \ldots, e_k$ without taking into account their sequential order along the path. Thus, $P^{(1)}(path_i) = P(e_1)P(e_2)P(e_3) \ldots P(e_k)$. In comparison, $P^{(2)}$ (see eq. 3) is a better candidate for identifying significant *strings*, as opposed to mere sets of nodes, because it takes into account the sequence of nodes along the path.

## 2.2 Step 2: identifying a significant pattern

Once the SPATH set is determined, the algorithm calculates the degree of cohesion $c_{ij}$ for each one of its member sub-paths, according to eq. 4. The $k$-gram matrix in eq. 4

accumulates all the statistics through order $k^{th} - 1$ of the SPATH embedded in the graph, with the zeroth order statistics located at the diagonal. The sub-path with the highest $c$-score is now tagged as a Significant Pattern.

Our experience shows that the two-stage mechanism just described induces coherent equivalence classes, leading to the formation of meaningful short patterns. The new pattern is added as a new vertex to the RDS graph, replacing the elements and edges it subsumes (Figure 1(b)). Note that only those edges of the multi-graph that belong to the detected pattern are rewired; edges that belong to sequences not subsumed by the pattern are left intact. This highly context-sensitive method of pattern abstraction, which is unique to our approach, allows ADIOS to achieve a high degree of representational parsimony without sacrificing generalization power.

$$P = \begin{pmatrix} p(e_1) & p(e_1|e_2) & p(e_1|e_2e_3) & ... & p(e_1|e_2e_3...e_k) \\ p(e_2|e_1) & p(e_2) & p(e_2|e_3) & ... & p(e_2|e_3e_4...e_k) \\ p(e_3|e_1e_2) & p(e_3|e_2) & p(e_3) & ... & p(e_3|e_4e_5...e_k) \\ \vdots & \vdots & \vdots & & \vdots \\ p(e_k|e_1e_2...e_{k-1}) & p(e_k|e_2e_3...e_{k-1}) & p(e_k|e_3e_4...e_{k-1}) & ... & p(e_k) \end{pmatrix}$$

$$c_{ij} = P_{ij} \log \frac{P_{ij}}{P_{i,j+1}} \qquad \text{for} \quad i > j \qquad (4)$$

During the pass over the corpus, the list of equivalence sets is updated continuously; new significant patterns are found using the *current* equivalence classes. For each set of candidate paths, the algorithm tries to fit one or more equivalence classes from the pool it maintains. Because an element can a appear in several classes, the algorithm must check different combinations of equivalence classes. The winner combination is always the largest class for which most of the members are found among the candidate paths in the set (the ratio between the number of members that have been found among the paths and the total number of members in the equivalence class is compared to a fixed threshold as one of the configuration acceptance criteria). When not all the members appear in an existing set, the algorithm creates a new equivalence class containing only those members that do. Thus, as the algorithm processes more and more text, it bootstraps itself and enriches the RDS graph structure with new SPs and their accompanying equivalence sets. The recursive nature of this process enables the algorithm to form more and more complex patterns, in a hierarchical manner.

The relationships among the distilled patterns can be visualized in a tree format, with tree depth corresponding to the level of recursion (e.g., Figure 2). Such a tree can be seen as a blueprint for creating acceptable ("grammatical") sequences of elements (strings). The number of all possible string configurations can be estimated and compared to the number of examples seen in the training corpus. The reciprocal of their ratio, $\eta$, is the generalization factor, which can be calculated for each pattern in the RDS graph (e.g., in Figure 1(c), $\eta = 0.33$). Patterns whose significance score $S$ and generalization factor $\eta$ are beneath certain thresholds are rejected. The algorithm halts if it processes a given amount of text without finding a new significant pattern or equivalence set (in real language acquisition this process may never stop).

## 2.3   The test module

A collection of patterns distilled from a corpus can be seen as a kind of empirically determined construction grammar; cf. [5], p.63. The patterns can eventually become highly abstract, thus endowing the model with an ability to generalize to unseen inputs. In pro-

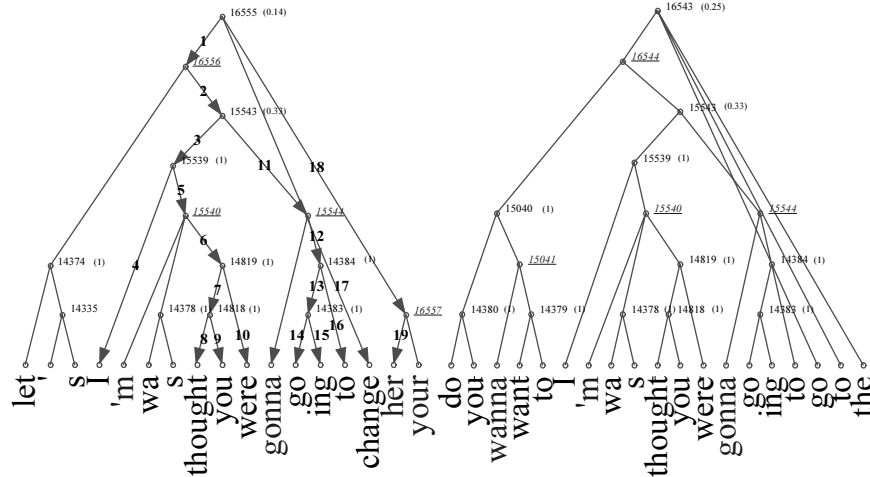

Figure 2: Two typical patterns extracted from a subset of the CHILDES collection [14]. Hundreds of such patterns and equivalence classes (underscored) together constitute a concise representation of the raw data. Some of the phrases that can be described/generated by patterns #16555 and #16543 are: let's change her...; I thought you gonna change her...; I was going to go to the.... None of these sentences appear in the training data, illustrating the ability of ADIOS to generalize. The numbers in parentheses denote the generalization factor $\eta$ of the patterns and their components (e.g., pattern #16555 generates $86\%$ new strings, while pattern #16543 generates $75\%$ new strings). The generation process, which operates as a depth-first search of the tree corresponding to a pattern, is illustrated on the left. For each non-terminal, the children are scanned from left to right; for each equivalence class (underscored), one member is chosen. The scan continues from the node corresponding to that member, with the elements reached at the terminal nodes being written out.

duction, generalization is possible, for example, when two equivalence classes are placed next to each other in a pattern, creating new paths among the members of the equivalence classes. In comprehension, generalization can also ensue from partial activation of existing patterns by novel inputs. This function is supported by the *test module*, designed to process a novel sentence by forming its distributed representation in terms of activities of existing patterns (a similar approach has been proposed for novel object and scene representation in vision [15]). These values, which can be used to support grammaticality judgment, are computed by propagating activation from bottom (the terminals) to top (the patterns) of the RDS. The initial activities $a_j$ of the terminals $e_j$ are calculated given the novel stimulus $s_1, \ldots, s_k$ as follows:

$$a_j = \max_{l=1..k} \left\{ P(s_l, e_j) \log \frac{P(s_l, e_j)}{P(s_l)P(e_j)} \right\} \tag{5}$$

where $P(s_l, e_j)$ is the joint probability of $s_l$ and $e_j$ appearing in the same equivalence class, and $P(s_l)$ and $P(e_j)$ are the probabilities of $s_l$ and $e_j$ appearing in any equivalence class. For an equivalence class, the value propagated upwards is the strongest non-zero activation of its members; for a pattern, it is the average weight of the children nodes, on the condition that all the children were activated by adjacent inputs. Activity propagation continues until it reaches the top nodes of the pattern lattice. When the algorithm encounters a novel word, all the members of the terminal equivalence class contribute a value of $\epsilon = 0.01$, which is then propagated upwards as usual. This enables the model to make an educated guess as to

the meaning of the unfamiliar word, by considering the patterns that become active.

## 3 Empirical results

### 3.1 Working with real data: the CHILDES' parents

To illustrate the scalability of our method, we describe here briefly the outcome of applying the PA algorithm to a subset of the CHILDES collection [14], which consists of transcribed speech produced by, or directed at, children. The corpus we selected contained 300,000 sentences (1.3 million tokens) produced by parents. The following results were derived from a snapshot of the algorithm's state after 14 real-time days. Working at a rate of 250 patterns per day, the algorithm identified 3400 patterns and 3200 equivalence classes, representing the corpus in terms of these elements. The outcome (for some examples, see Figure 2) was encouraging: the algorithm found intuitively significant SPs and produced semantically adequate corresponding equivalence sets. The algorithm's considerable ability to recombine and reuse constructions it learns is illustrated by the following examples, in which a few of the sentences generated by ADIOS (left) are shown alongside sentences from CHILDES described by the same compositions of patterns:

| *ADIOS* | *CHILDES (parents' speech)* |
|---|---|
| what doe s Spot say ? | where doe s it go ? |
| I don 't think it ' s good ! | that ' s good ! |
| it ' s gon ta go first . | dog ' s gon ta eat first . |
| there ' s a cup and there ' s some lamb s . | there ' s a table and there ' s some chair s . |

### 3.2 Novel inputs

We have assessed the ability of the ADIOS model to deal with novel inputs by training it on the CHILDES collection and then subjecting it to a grammaticality judgment test, in the form of multiple choice questions used in English as Second Language (ESL) classes. The particular test (http://www.forumeducation.net/servlet/pages/vi/mat/gram/dia001.htm) has been administered to more than $10,000$ people in the Göteborg (Sweden) education system as a diagnostic tool when assessing students on upper secondary levels (that is, children who typically had 9 years of school, but only 6-7 years of English; a test designed for assessing proficiency of younger subjects in their native language would be more suitable, but is not available). The test consists of 100 three-choice questions; a score lower than $50\%$ is considered pre-intermediate, $50\% - 70\%$ intermediate, and a score greater than $70\%$ – advanced, with $65\%$ being the average score for the population mentioned. For each of the three choices in a given question, our algorithm provided a grammaticality score. The choice with the highest score was declared as the winner; if two choices received the same top score, the answer was "don't know". The algorithm's performance in this test at different stages of learning is plotted in Figure 3 versus the number of corpus sentences that have been processed. Over the course of training, the proportion of questions that received a definite answer grew (solid curve), while the proportion of correct answers remained around $60\%$ (dashed curve).

The best results were achieved with the ensemble of patterns distilled from two separate runs (two different generalization factors were applied in each run: 0.01 and 0.05). As a benchmark, we compared the performance of ADIOS in this test with that of a word bi-gram model. The latter was tested using the same procedure as ADIOS, except that significant patterns in the bi-gram model were defined as all the word pairs in the corpus (we emphasize that there is no training phase in the bi-gram model, as all the "patterns" are already available in the raw data). ADIOS outperformed the bi-gram model by answering $60\%$ of the questions with $60\%$ hits, compared to $20\%$ of the questions with only $45\%$ hits

for the latter (note that chance performance in this test is 33%).

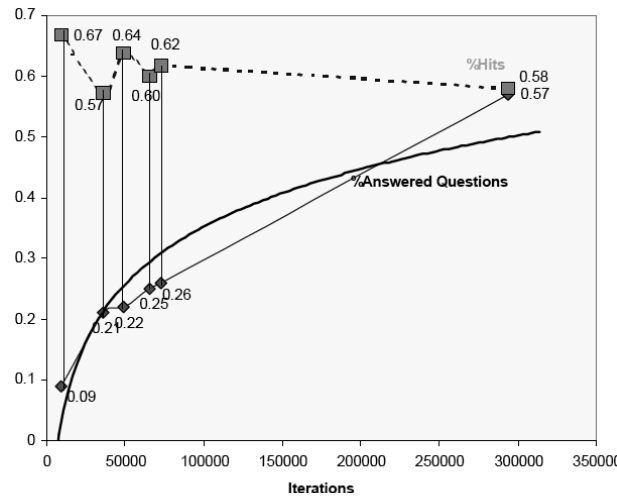

Figure 3: The performance of ADIOS2 in an ESL test based on grammaticality judgment, plotted against the number of sentences (paths) scanned during training. The solid curve represents the percentage of questions with a valid answer; the dashed curve shows the percentage of correct answers.

## 4   Concluding remarks

The ADIOS model incrementally learns the (morpho)syntax of English from "raw" input by distilling structural regularities (which can be thought of as constructions [16, 4]) from the accrued statistical co-occurrence and contextual cues. The resulting pattern-based representations are more powerful than finite automata because of their potential for recursion. Their depth, however, is not unbounded (rather, it is driven by the demands of the training data), a limitation that actually makes ADIOS a better candidate model for psycholinguistics (cf. the human limitations on processing recursion [17]). The patterns learned by ADIOS are also more powerful than context-free rewriting rules, because of their conservative nature: members of an equivalence class are only ever considered as interchangeable in a specific context, a characteristic that distinguishes ADIOS from related approaches [18, 10, 9]. On the one hand, this results in larger – but not unmanageable – demands on memory (more patterns need to be stored); on the other hand, crucially, it leads to efficient unsupervised probabilistic learning, and subsequent judicious use, of linguistic knowledge.

The ultimate goal of this project is to address the entire spectrum of English syntax-related phenomena (and, eventually, semantics, which, as the construction grammarians hold, is intimately connected to syntax [16, 4]). With respect to some of these, the ADIOS model is already known to behave reasonably: for example, subject-verb agreement (even long-range) is captured properly, due to the conservative structured pattern abstraction. While providing empirical evidence that can be brought to bear on the poverty of the stimulus argument for innateness, our work does not, of course, resolve completely the outstanding issues. In particular, the treatment of many aspects of syntax such as anaphora, auxiliaries, *wh*-questions, passive, control, etc. [19], awaits both further computational experimentation and further theoretical work.

*Acknowledgments.* Supported by the US-Israel Binational Science Foundation, the Dan David Prize Foundation, and the Horowitz Center for Complexity Science. We thank Todd

Siegel for helpful suggestions.

## References

[1] N. Chomsky. *Knowledge of language: its nature, origin, and use*. Praeger, New York, 1986.

[2] S. Pinker. *The Language Instinct: How the Mind Creates Language.* William Morro, New York, NY, 1994.

[3] P. J. Hopper. Emergent grammar. In M. Tomasello, editor, *The new psychology of language*, pp. 155–175. Erlbaum, Mahwah, NJ, 1998.

[4] W. Croft. *Radical Construction Grammar: syntactic theory in typological perspective.* Oxford University Press, Oxford, 2001.

[5] R. W. Langacker. *Foundations of cognitive grammar*, volume I: theoretical prerequisites. Stanford University Press, Stanford, CA, 1987.

[6] A. Wray. *Formulaic language and the lexicon*. Cambridge University Press, Cambridge, UK, 2002.

[7] K. Lari and S. J. Young. The estimation of stochastic context-free grammars using the Inside-Outside algorithm. *Computer Speech and Language*, 4:35–56, 1990.

[8] F. Pereira and Y. Schabès. Inside-Outside reestimation from partially bracketed corpora. In *Annual Meeting of the ACL*, pp. 128–135, 1992.

[9] D. Klein and C. D. Manning. Natural language grammar induction using a constituent-context model. In T. G. Dietterich, S. Becker, and Z. Ghahramani, ed., *Advances in Neural Information Processing Systems 14*, Cambridge, MA, 2002. MIT Press.

[10] M. van Zaanen and P. Adriaans. Comparing two unsupervised grammar induction systems: Alignment-based learning vs. EMILE. Report 05, School of Computing, Leeds University, 2001.

[11] M. Gross. The construction of local grammars. In E. Roche and Y. Schabès, ed., *Finite-State Language Processing*, pp. 329–354. MIT Press, Cambridge, MA, 1997.

[12] J. G. Wolff. Learning syntax and meanings through optimization and distributional analysis. In Y. Levy, I. M. Schlesinger, and M. D. S. Braine, ed., *Categories and Processes in Language Acquisition*, pp. 179–215. Lawrence Erlbaum, Hillsdale, NJ, 1988.

[13] Z. Solan, E. Ruppin, D. Horn, and S. Edelman. Automatic acquisition and efficient representation of syntactic structures. In S. Thrun, editor, *Advances in Neural Information Processing*, volume 15, Cambridge, MA, 2003. MIT Press.

[14] B. MacWhinney and C. Snow. The child language exchange system. *Journal of Computational Lingustics*, 12:271–296, 1985.

[15] S. Edelman. Constraining the neural representation of the visual world. *Trends in Cognitive Sciences*, 6:125–131, 2002.

[16] A. E. Goldberg. *Constructions: A construction grammar approach to argument structure*. University of Chicago Press, Chicago, 1995.

[17] M. C. MacDonald and M. H. Christiansen. Reassessing working memory: A comment on Just and Carpenter (1992) and Waters and Caplan (1996). *Psychological Review*, 109:35–54, 2002.

[18] A. Clark. *Unsupervised Language Acquisition: Theory and Practice*. PhD thesis, COGS, University of Sussex, 2001.

[19] I. A. Sag and T. Wasow. *Syntactic theory: a formal introduction*. CSLI Publications, Stanford, CA, 1999.
